# Hierarchical Apprenticeship Learning, with Application to Quadruped Locomotion

**J. Zico Kolter, Pieter Abbeel, Andrew Y. Ng**
Department of Computer Science
Stanford University
Stanford, CA 94305
{kolter, pabbeel, ang}@cs.stanford.edu

## Abstract

We consider apprenticeship learning—learning from expert demonstrations—in the setting of large, complex domains. Past work in apprenticeship learning requires that the expert demonstrate complete trajectories through the domain. However, in many problems even an expert has difficulty controlling the system, which makes this approach infeasible. For example, consider the task of teaching a quadruped robot to navigate over extreme terrain; demonstrating an optimal policy (i.e., an optimal set of foot locations over the entire terrain) is a highly non-trivial task, even for an expert. In this paper we propose a method for *hierarchical apprenticeship learning*, which allows the algorithm to accept isolated advice at different hierarchical levels of the control task. This type of advice is often feasible for experts to give, even if the expert is unable to demonstrate complete trajectories. This allows us to extend the apprenticeship learning paradigm to much larger, more challenging domains. In particular, in this paper we apply the hierarchical apprenticeship learning algorithm to the task of quadruped locomotion over extreme terrain, and achieve, to the best of our knowledge, results superior to any previously published work.

## 1 Introduction

In this paper we consider *apprenticeship learning* in the setting of large, complex domains. While most reinforcement learning algorithms operate under the Markov decision process (MDP) formalism (where the reward function is typically assumed to be given a priori), past work [1, 13, 11] has noted that often the reward function itself is difficult to specify by hand, since it must quantify the trade off between many features. Apprenticeship learning is based on the insight that often it is easier for an "expert" to demonstrate the desired behavior than it is to specify a reward function that induces this behavior. However, when attempting to apply apprenticeship learning to large domains, several challenges arise. First, past algorithms for apprenticeship learning require the expert to demonstrate complete trajectories in the domain, and we are specifically concerned with domains that are sufficiently complex so that even this task is not feasible. Second, these past algorithms require the ability to solve the "easier" problem of finding a nearly optimal policy *given* some candidate reward function, and even this is challenging in large domains. Indeed, such domains often necessitate hierarchical control in order to reduce the complexity of the control task [2, 4, 15, 12].

As a motivating application, consider the task of navigating a quadruped robot (shown in Figure 1(a)) over challenging, irregular terrain (shown in Figure 1(b,c)). In a naive approach, the dimensionality of the state space is prohibitively large: the robot has 12 independently actuated joints, and the state must also specify the current three-dimensional position and orientation of the robot, leading to an 18-dimensional state space that is well beyond the capabilities of standard RL algorithms. Fortunately, this control task succumbs very naturally to a hierarchical decomposition: we first plan a general path over the terrain, then plan footsteps along this path, and finally plan joint movements

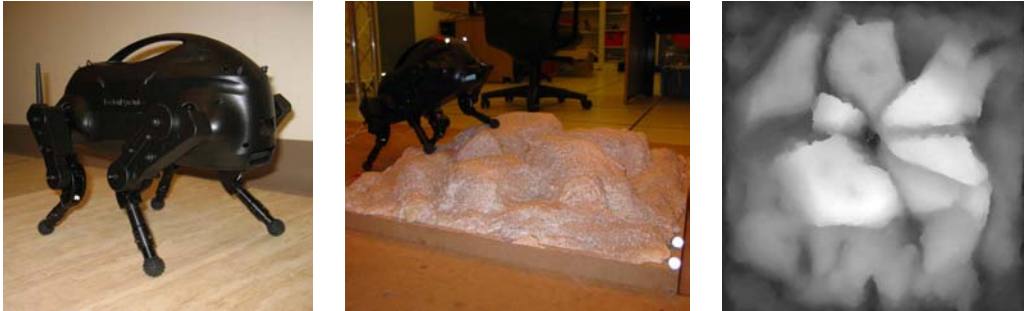

Figure 1: (a) LittleDog robot, designed and built by Boston Dynamics, Inc. (b) Typical terrain. (c) Height map of the depicted terrain. (Black = 0cm altitude, white = 12cm altitude.)

to achieve these footsteps. However, it is very challenging to specify a proper reward, specifically for the higher levels of control, as this requires quantifying the trade-off between many features, including progress toward a goal, the height differential between feet, the slope of the terrain underneath its feet, etc. Moreover, consider the apprenticeship learning task of specifying a complete set of foot locations, across an entire terrain, that properly captures all the trade-offs above; this itself is a highly non-trivial task.

Motivated by these difficulties, we present a unified method for *hierarchical apprenticeship learning*. Our approach is based on the insight that, while it may be difficult for an expert to specify entire optimal trajectories in a large domain, it is much easier to "teach hierarchically": that is, if we employ a hierarchical control scheme to solve our problem, it is much easier for the expert to give advice independently at each level of this hierarchy. At the lower levels of the control hierarchy, our method only requires that the expert be able to demonstrate good *local* behavior, rather than behavior that is optimal for the entire task. This type of advice is often feasible for the expert to give even when the expert is entirely unable to give full trajectory demonstrations. Thus the approach allows for apprenticeship learning in extremely complex, previously intractable domains.

The contributions of this paper are twofold. First, we introduce the hierarchical apprenticeship learning algorithm. This algorithm extends the apprenticeship learning paradigm to complex, high-dimensional control tasks by allowing an expert to demonstrate desired behavior at multiple levels of abstraction. Second, we apply the hierarchical apprenticeship approach to the quadruped locomotion problem discussed above. By applying this method, we achieve performance that is, to the best of our knowledge, well beyond any published results for quadruped locomotion.[1]

The remainder of this paper is organized as follows. In Section 2 we discuss preliminaries and notation. In Section 3 we present the general formulation of the hierarchical apprenticeship learning algorithm. In Section 4 we present experimental results, both on a hierarchical multi-room grid world, and on the real-world quadruped locomotion task. Finally, in Section 5 we discuss related work and conclude the paper.

## 2 Preliminaries and Notation

A Markov decision process (MDP) is a tuple $(S, A, T, H, D, R)$, where $S$ is a set of states; $A$ is a set of actions, $T = \{P_{sa}\}$ is a set of state transition probabilities (here, $P_{sa}$ is the state transition distribution upon taking action $a$ in state $s$); $H$ is the horizon which corresponds to the number of time-steps considered; $D$ is a distribution over initial states; and $R : S \to \mathbb{R}$ is a reward function. As we are often concerned with MDPs for which no reward function is given, we use the notation MDP\R to denote an MDP minus the reward function. A policy $\pi$ is a mapping from states to a probability distribution over actions. The value of a policy $\pi$ is given by $V(\pi) = E\left[\sum_{t=0}^{H} R(s_t)|\pi\right]$, where the expectation is taken with respect to the random state sequence $s_0, s_1, \ldots, s_H$ drawn by stating from the state $s_0$ (drawn from distribution $D$) and picking actions according to $\pi$.

Often the reward function $R$ can be represented more compactly as a function of the state. Let $\phi : S \to \mathbb{R}^n$ be a mapping from states to a set of features. We consider the case where the reward function $R$ is a linear combination of the features: $R(s) = w^T \phi(s)$ for parameters $w \in \mathbb{R}^n$. Then we have that the value of a policy $\phi$ is linear in the reward function weights

$$V(\pi) = E[\textstyle\sum_{t=0}^{H} R(s_t)|\pi] = E[\textstyle\sum_{t=0}^{H} w^T \phi(s_t)|\pi] = w^T E[\textstyle\sum_{t=0}^{H} \phi(s_t)|\pi] = w^T \mu_\phi(\pi) \quad (1)$$

where we used linearity of expectation to bring $w$ outside of the expectation. The last quantity defines the vector of *feature expectations* $\mu_\phi(\pi) = E[\sum_{t=0}^{H} \phi(s_t)|\pi]$.

## 3   The Hierarchical Apprenticeship Learning Algorithm

We now present our hierarchical apprenticeship learning algorithm (hereafter HAL). For simplicity, we present a *two level* hierarchical formulation of the control task, referred to generically as the *low-level* and *high-level* controllers. The extension to higher order hierarchies poses no difficulties.

### 3.1   Reward Decomposition in HAL

At the heart of the HAL algorithm is a simple decomposition of the reward function that links the two levels of control. Suppose that we are given a hierarchical decomposition of a control task in the form of two MDP\Rs — a low-level and a high-level MDP\R, denoted $M_\ell = (S_\ell, A_\ell, T_\ell, H_\ell, D_\ell)$ and $M_h = (S_h, A_h, T_h, H_h, D_h)$ respectively — and a partitioning function $\psi : S_\ell \to S_h$ that maps low level states to high-level states (the assumption here is that $|S_h| \ll |S_\ell|$ so that this hierarchical decomposition actually provides a computational gain).[2] For example, in the case of the quadruped locomotion problem the low-level MDP\R describes the state of all four feet, while the high-level MDP\R describes only the position of the robot's center of mass. As is standard in apprenticeship learning, we suppose that the rewards in the low-level MDP\R can be represented as a linear function of state features, $R(s_\ell) = w^T \phi(s_\ell)$. The HAL algorithm assumes that the reward of a high-level state is equal to the average reward over all its corresponding low-level states. Formally

$$R(s_h) = \frac{1}{N(s_h)} \sum_{s_\ell \in \psi^{-1}(s_h)} R(s_\ell) = \frac{1}{N(s_h)} \sum_{s_\ell \in \psi^{-1}(s_h)} w^T \phi(s_\ell) = \frac{1}{N(s_h)} w^T \sum_{s_\ell \in \psi^{-1}(s_h)} \phi(s_\ell)$$

$$(2)$$

where $\psi^{-1}(s_h)$ denotes the inverse image of the partitioning function and $N(s_h) = |\psi^{-1}(s_h)|$. While this may not always be the most ideal decomposition of the reward function in many cases— for example, we may want to let the reward of a high-level state be the *maximum* of its low level state rewards to capture the fact that an ideal agent would always seek to maximize reward at the lower level, or alternatively the *minimum* of its low level state rewards to be robust to worst-case outcomes—it captures the idea that in the absence of other prior information, it seems reasonable to assume a uniform distribution over the low-level states corresponding to a high-level state. An important consequence of (2) is that the high level reward is now also linear in the low-level reward weights $w$. This will enable us in the subsequent sections to formulate a unified hierarchical apprenticeship learning algorithm that is able to incorporate expert advice at both the high level and the low level simultaneously.

### 3.2   Expert Advice at the High Level

Similar to past apprenticeship learning methods, expert advice at the high level consists of full policies demonstrated by the expert. However, because the high-level MDP\R can be significantly simpler than the low-level MDP\R, this task can be substantially easier. If the expert suggests that $\pi_{h,E}^{(i)}$ is an optimal policy for some given MDP\R $M_h^{(i)}$, then this corresponds to the following constraint, which states that the expert's policy outperforms all other policies:

$$V^{(i)}(\pi_{h,E}^{(i)}) \geq V^{(i)}(\pi_h^{(i)}) \quad \forall \pi_h^{(i)}.$$

Equivalently, using (1), we can formulate this constraint as follows:

$$w^T \mu_\phi^{(i)}(\pi_{h,E}^{(i)}) \geq w^T \mu_\phi(\pi_h^{(i)}) \quad \forall \pi_h^{(i)}.$$

While we may not be able to obtain the exact feature expectations of the expert's policy if the high-level transitions are stochastic, observing a single expert demonstration corresponds to receiving

a sample from these feature expectations, so we simply use the observed expert features counts $\hat{\mu}_\phi^{(i)}(\pi_{h,E}^{(i)})$ in lieu of the true expectations. By standard sample complexity arguments [1], it can be shown that a sufficient number of observed feature counts will converge to the true expectation. To resolve the ambiguity in $w$, and to allow the expert to provide noisy advice, we use regularization and slack variables (similar to standard SVM formulations), which results in the following formulation:

$$\min_{w,\eta} \quad \frac{1}{2}\|w\|_2^2 + C_h \sum_{i=1}^n \eta^{(i)}$$
$$\text{s.t.} \quad w^T \hat{\mu}_\phi^{(i)}(\pi_{h,E}^{(i)}) \geq w^T \mu_\phi(\pi_h^{(i)}) + 1 - \eta^{(i)} \quad \forall \pi_h^{(i)}, i$$

where $\pi_h^{(i)}$ indexes over all high-level policies, $i$ indexes over all MDPs, and $C_h$ is a regularization constant.[3] Despite the fact that there are an exponential number of possible policies there are well-known algorithms that are able to solve this optimization problem; however, we defer this discussion until after presenting our complete formulation.

### 3.3 Expert Advice at the Low Level

Our approach differs from standard apprenticeship learning when we consider advice at the low level. Unlike the apprenticeship learning paradigm where an expert specifies full trajectories in the target domain, we allow for an expert to specify single, greedy actions in the low-level domain. Specifically, if the agent is in state $s_\ell$ and the expert suggests that the best greedy action would move to state $s'_\ell$, this corresponds directly to a constraint on the *reward* function, namely that

$$R(s'_\ell) \geq R(s''_\ell)$$

for all other states $s''_\ell$ that can be reached from the current state (we say that $s''_\ell$ is "reachable" from the current state $s_\ell$ if $\exists a \text{ s.t.} P_{s_\ell a}(s''_\ell) > \epsilon$ for some $0 < \epsilon \leq 1$).[4] This results in the following constraints on the reward function parameters $w$,

$$w^T \phi(s'_\ell) \geq w^T \phi(s''_\ell)$$

for all $s''_\ell$ reachable from $s_\ell$. As before, to resolve the ambiguity in $w$ and to allow for the expert to provide noisy advice, we use regularization and slack variables. This gives:

$$\min_{w,\xi} \quad \frac{1}{2}\|w\|_2^2 + C_\ell \sum_{j=1}^m \xi^{(j)}$$
$$\text{s.t.} \quad w^T \phi(s'^{(j)}_\ell) \geq w^T \phi(s''^{(j)}_\ell) + 1 - \xi^{(j)} \quad \forall s''^{(j)}_\ell, j$$

where $s''^{(j)}_\ell$ indexes over all states reachable from $s'^{(j)}_\ell$ and $j$ indexes over all low-level demonstrations provided by the expert.

### 3.4 The Unified HAL Algorithm

From (2) we see the high level and low level rewards are a linear combination of the same set of reward weights $w$. This allows us to combine both types of expert advice presented above to obtain the following unified optimization problem

$$\min_{w,\eta,\xi} \quad \frac{1}{2}\|w\|_2^2 + C_\ell \sum_{j=1}^m \xi^{(j)} + C_h \sum_{i=1}^n \eta^{(i)}$$
$$\text{s.t.} \quad w^T \phi(s'^{(j)}_\ell) \geq w^T \phi(s''^{(j)}_\ell) + 1 - \xi^{(j)} \quad \forall s''^{(j)}_\ell, j \qquad (3)$$
$$w^T \hat{\mu}_\phi^{(i)}(\pi_{h,E}^{(i)}) \geq w^T \mu_\phi(\pi_h^{(i)}) + 1 - \eta^{(i)} \quad \forall \pi_h^{(i)}, i.$$

This optimization problem is convex, and can be solved efficiently. In particular, even though the optimization problem has an exponentially large number of constraints (one constraint per policy), the optimum can be found efficiently (i.e., in polynomial time) using, for example, the ellipsoid method, since we can efficiently identify a constraint that is violated.[5] However, in practice we found the following constraint generation method more efficient:

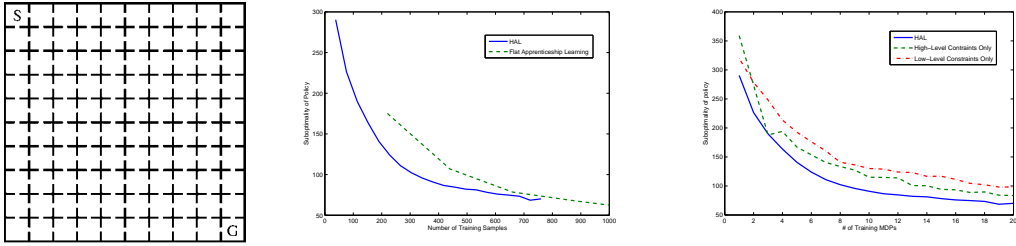

Figure 2: (a) Picture of the multi-room gridworld environment. (b) Performance versus number of training samples for HAL and flat apprenticeship learning. (c) Performance versus number of training MDPs for HAL versus using only low-level or only high-level constraints.

1. Begin with no expert path constraints.
2. Find the current reward weights by solving the current optimization problem.
3. Solve the reinforcement learning problem at the high level of the hierarchy to find the optimal (high-level) policies for the current reward for each MDP\R, $i$. If the optimal policy violates the current (high level) constraints, then add this constraint to the current optimization problem and goto Step (2). Otherwise, no constraints are violated and the current reward weights are the solution of the optimization problem.

## 4 Experimental Results

### 4.1 Gridworld

In this section we present results on a multi-room gridworld domain with unknown cost. While this is not meant to be a challenging control task, it allows us to compare the performance of HAL to traditional "flat" (non-hierarchical) apprenticeship learning methods, as these algorithms are feasible in such domains. The grid world domain has a very natural hierarchical decomposition: if we average the cost over each room, we can form a "high-level" approximation of the grid world. Our hierarchical controller first plans in this domain to choose a path over the rooms. Then for each room along this path we plan a low-level path to the desired exit.

Figure 2(b) shows the performance versus number of training examples provided to the algorithm (where one training example equals one action demonstrated by the expert).[6] As expected, the flat apprenticeship learning algorithm eventually converges to a superior policy, since it employs full value iteration to find the optimal policy, while HAL uses the (non-optimal) hierarchical controller. However, for small amounts of training data, HAL outperforms the flat method, since it is able to leverage the small amount of data provided by the expert at both levels of the hierarchy. Figure 2(c) shows performance versus number of MDPs in the training set for HAL and well as for algorithms which receive the same training data as HAL (that is, both high level and low level expert demonstrations), but which make use of only one or the other. Here we see that HAL performs substantially better. This is not meant to be a direct comparison of the different methods, since HAL obtains more training data per MDP than the single-level approaches. Rather, this experiment illustrates that in situations where one has access to both high-level and low-level advice, it is advantageous to use

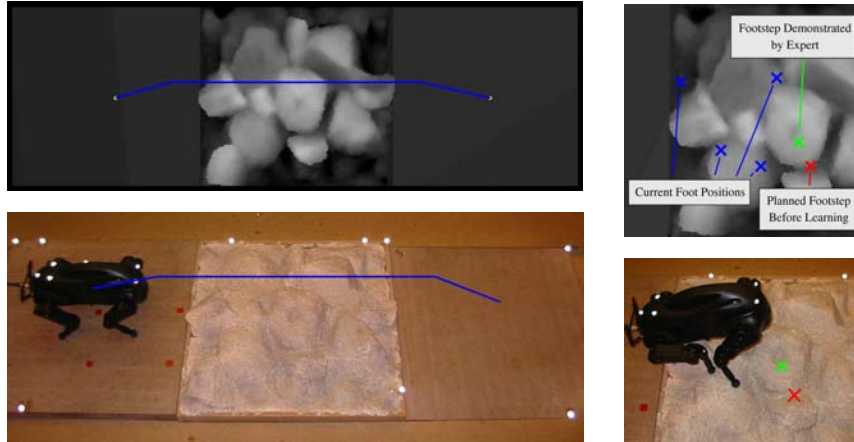

Figure 3: (a) High-level (path) expert demonstration. (b) Low-level (footstep) expert demonstration.

both. This will be especially important in domains such as the quadruped locomotion task, where we have access to very few training MDPs (i.e., different terrains).

## 4.2 Quadruped Robot

In this section we present the primary experimental result of this paper, a successful application of hierarchical apprenticeship learning to the task of quadruped locomotion. Videos of the results in this section are available at ***http://cs.stanford.edu/~kolter/nips07videos***.

### 4.2.1 Hierarchical Control for Quadruped Locomotion

The LittleDog robot, shown in Figure 1, is designed and built by Boston Dynamics, Inc. The robot consists of 12 independently actuated servo motors, three on each leg, with two at the hip and one at the knee. It is equipped with an internal IMU and foot force sensors. We estimate the robot's state using a motion capture system that tracks reflective markers on the robot's body. We perform all computation on a desktop computer, and send commands to the robot via a wireless connection.

As mentioned in the introduction, we employ a hierarchical control scheme for navigating the quadruped over the terrain. Due to space constraints, we describe the complete control system briefly, but a much more detailed description can be found in [8]. The high level controller is a *body path planner*, that plans an approximate trajectory for the robot's center of mass over the terrain; the low-level controller is a *footstep planner* that, given a path for the robot's center, plans a set of footsteps that follow this path. The footstep planner uses a reward function that specifies the relative trade-off between several different features of the robot's state, including (i) several features capturing the roughness and slope of the terrain at several different spatial scales around the robot's feet, (ii) distance of the foot location from the robot's desired center, (iii) the area and inradius of the support triangle formed by the three stationary feet, and other similar features. Kinematic feasibility is required for all candidate foot locations and collision of the legs with obstacles is forbidden. To form the high-level cost, we aggregate features from the footstep planner. In particular, for each foot we consider all the footstep features within a 3 cm radius of the foot's "home" position (the desired position of the foot relative to the center of mass in the absence of all other discriminating features), and aggregate these features to form the features for the body path planner. While this is an approximation, we found that it performed very well in practice, possibly due to its ability to account for stochasticity of the domain. After forming the cost function for both levels, we used value iteration to find the optimal policy for the body path planner, and a five-step lookahead receding horizon search to find a good set of footsteps for the footstep planner.

### 4.2.2 Hierarchical Apprenticeship Learning for Quadruped Locomotion

All experiments were carried out on two terrains: a relatively easy terrain for training, and a significantly more challenging terrain for testing. To give advice at the high level, we specified complete body trajectories for the robot's center of mass, as shown in Figure 3(a). To give advice for the low level we looked for situations in which the robot stepped in a suboptimal location, and then indicated the correct greedy foot placement, as shown in Figure 3(b). The entire training set con-

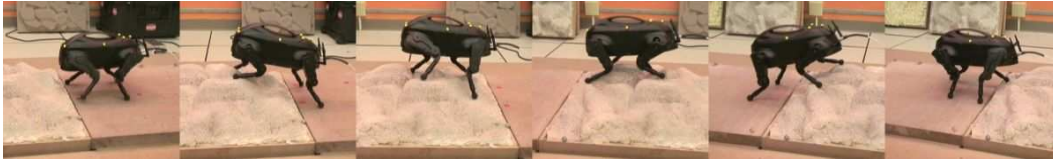

Figure 4: Snapshots of quadruped while traversing the testing terrain.

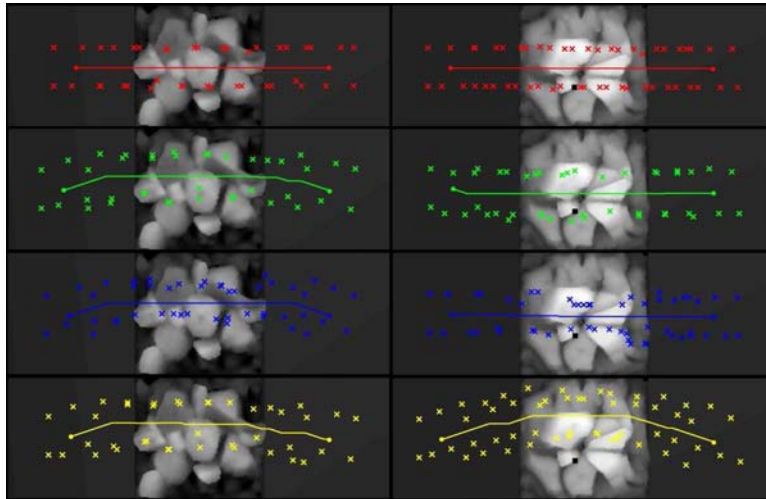

Figure 5: Body and footstep plans for different constraints on the training (left) and testing (right) terrains: (Red) No Learning, (Green) HAL, (Blue) Path Only, (Yellow) Footstep Only.

sisted of a single high-level path demonstration across the training terrain, and 20 low-level footstep demonstrations on this terrain; it took about 10 minutes to collect the data.

Even from this small amount of training data, the learned system achieved excellent performance, not only on the training board, but also on the much more difficult testing board. Figure 4 shows snapshots of the quadruped crossing the testing board. Figure 5 shows the resulting footsteps taken for each of the different types of constraints, which shows a very large qualitative difference between the footsteps chosen before and after training. Table 1 shows the crossing times for each of the different types of constraints. As shown, he HAL algorithm outperforms all the intermediate methods. Using only footstep constraints does quite well on the training board, but on the testing board the lack of high-level training leads the robot to take a very roundabout route, and it performs much worse. The quadruped fails at crossing the testing terrain when learning from the path-level demonstration only or when not learning at all.

Finally, prior to undertaking our work on hierarchical apprenticeship learning, we invested several weeks attempting to hand-tune controller capable of picking good footsteps across challenging terrain. However, none of our previous efforts could significantly outperform the controller presented here, learned from about 10 minutes worth of data, and many of our previous efforts performed substantially worse.

## 5   Related Work and Discussion

The work presented in this paper relates to many areas of reinforcement learning, including apprenticeship learning and hierarchical reinforcement learning, and to a large body of past work in quadruped locomotion. In the introduction and in the formulation of our algorithm we discussed the connection to the inverse reinforcement learning algorithm of [1] and the maximum margin planning algorithm of [13]. In addition, there has been subsequent work [14] that extends the maximum margin planning framework to allow for the automated addition of new features through a boosting procedure; There has also been much recent work in reinforcement learning on hierarchical reinforcement learning; a recent survey is [2]. However, all the work in this area that we are aware of deals with the more standard reinforcement learning formulation where known rewards are given to the agent as it acts in a (possibly unknown) environment. In contrast, our work follows the apprenticeship learning paradigm where the model, but not the rewards, are known to the agent. Prior work on legged locomotion has mostly focused on generating gaits for stably traversing fairly flat

|          |            | HAL   | Feet Only | Path Only | No Learning |
|----------|------------|-------|-----------|-----------|-------------|
| **Training** | Time (sec) | 31.03 | 33.46     | —         | 40.25       |
| **Testing**  | Time (sec) | 35.25 | 45.70     | —         | —           |

Table 1: Execution times for different constraints on training and testing terrains. Dashes indicate that the robot fell over and did not reach the goal.

terrain (see, among many others, [10], [7]). Only very few learning algorithms, which attempt to generalize to previously unseen terrains, have been successfully applied before [6, 3, 9]. The terrains considered in this paper go well beyond the difficulty level considered in prior work.

## 6  Acknowledgements

We gratefully acknowledge the anonymous reviewers for helpful suggestions. This work was supported by the DARPA Learning Locomotion program under contract number FA8650-05-C-7261.

## Footnotes

[1]There are several other institutions working with the LittleDog robot, and many have developed (unpublished) systems that are also very capable. As of the date of submission, we believe that the controller presented in this paper is on par with the very best controllers developed at other institutions. For instance, although direct comparison is difficult, the fastest running time that any team achieved during public evaluations was 39 seconds. In Section 4 we present results crossing terrain of comparable difficulty and distance in 30-35 seconds.

[2]As with much work in reinforcement learning, it is the assumption of this paper that the hierarchical decomposition of a control task is *given* by a system designer. While there has also been recent work on the automated discovery of state abstractions[5], we have found that there is often a very natural decomposition of control tasks into multiple levels (as we will discuss for the specific case of quadruped locomotion).

[3] This formulation is not entirely correct by itself, due to the fact that it is impossible to separate a policy from *all* policies (including itself) by a margin of one, and so the exact solution to this problem will be $w = 0$. To deal with this, one typically scales the margin or slack by some loss function that quantifies how different two policies are [16, 17], and this is the approach taken by Ratliff, et al. [13] in their maximum margin planning algorithm. Alternatively, Abbeel & Ng [1], solve the optimization problem without any slack, and notice that as soon as the problem becomes infeasible, the expert's policy lies in the convex hull of the generated policies. However, in our full formulation with low-level advice also taken into account, this becomes less of an issue, and so we present the above formulation for simplicity. In all experiments where we use only the high-level constraints, we employ margin scaling as in [13].

[4] Alternatively, one interpret low-level advice at the level of *actions*, and interpret the expert picking action $a$ as the constraint that $\sum_{s'} P_{sa}(s')R(s') \geq \sum_{s'} P_{sa'}(s')R(s') \ \forall a' \neq a$. However, in the domains we consider, where there is a clear set of "reachable" states from each state, the formalism above seems more natural.

[5] Similar techniques are employed by [17] to solve structured prediction problems. Alternatively, Ratliff, et al. [13] take a different approach, and move the constraints into the objective by eliminating the slack variables, then employ a subgradient method.

[6]Experimental details: We consider a 111x111 grid world, evenly divided into 100 rooms of size 10x10 each. There are walls around each room, except for a door of size 2 that connects a room to each of its neighbors (a picture of the domain is shown in figure 2(a)). Each state has 40 binary features, sampled from a distribution particular to that room, and the reward function is chosen randomly to have 10 "small" [-0.75, -0.25], negative rewards, 20 "medium" [-1.0 -2.0] negative rewards, and 10 "high" [-3.0 -5.0] negative rewards. In all cases we generated multiple training MDPs, which differ in which features are active at each state and we provided the algorithm with one expert demonstration for each sampled MDP. After training on each MDP we tested on 25 holdout MDPs generated by the same process. In all cases the results were averaged over 10 runs. For all our experiments, we fixed the ratio of $C_h/C_\ell$ so that the both constraints were equally weighted (i.e., if it typically took $t$ low level actions to accomplish one high-level action, then we used a ratio of $C_h/C_\ell = t$). Given this fixed scaling, we found that the algorithm was generally insensitive (in terms of the resulting policy's suboptimality) to scaling of the slack penalties. In the comparison of HAL with flat apprenticeship learning in Figure 2(b), one training example corresponds to one expert action. Concretely, for HAL the number of training examples for a given training MDP corresponds to the number of high level actions in the high level demonstration plus the (equal) number of low level expert actions provided. For flat apprenticeship learning the number of training examples for a given training MDP corresponds to the number of expert actions in the expert's full trajectory demonstration.

## References

[1] Pieter Abbeel and Andrew Y. Ng. Apprenticeship learning via inverse reinforcement learning. In *Proceedings of the International Conference on Machine Learning*, 2004.

[2] Andrew G. Barto and Sridhar Mahadevan. Recent advances in hierarchical reinforcement learning. *Discrete Event Dynamic Systems: Theory and Applications*, 13:41–77, 2003.

[3] Joel Chestnutt, James Kuffner, Koichi Nishiwaki, and Satoshi Kagami. Planning biped navigation strategies in complex environments. In *Proceedings of the International Conference on Humanoid Robotics*, 2003.

[4] Thomas G. Dietterich. Hierarchical reinforcement learning with the MAXQ value function decomposition. *Journal of Artificial Intelligence Research*, 13:227–303, 2000.

[5] Nicholas K. Jong and Peter Stone. State abstraction discovery from irrelevant state variables. In *Proceedings of the International Joint Conference on Artificial Intelligence*, 2005.

[6] H. Kim, T. Kang, V. G. Loc, and H. R. Choi. Gait planning of quadruped walking and climbing robot for locomotion in 3D environment. In *Proceedings of the International Conference on Robotics and Automation*, 2005.

[7] Nate Kohl and Peter Stone. Machine learning for fast quadrupedal locomotion. In *Proceedings of AAAI*, 2004.

[8] J. Zico Kolter, Mike P. Rodgers, and Andrew Y. Ng. A complete control architecture for quadruped locomotion over rough terrain. In *Proceedings of the International Conference on Robotics and Automation (to appear)*, 2008.

[9] Honglak Lee, Yirong Shen, Chih-Han Yu, Gurjeet Singh, and Andrew Y. Ng. Quadruped robot obstacle negotiation via reinforcement learning. In *Proceedings of the International Conference on Robotics and Automation*, 2006.

[10] Jun Morimoto and Christopher G. Atkeson. Minimax differential dynamic programming: An application to robust biped walking. In *Neural Information Processing Systems 15*, 2002.

[11] Gergeley Neu and Csaba Szepesvári. Apprenticeship learning using inverse reinforcement learning and gradient methods. In *Proceedings of Uncertainty in Artificial Intelligence*, 2007.

[12] Ronald Parr and Stuart Russell. Reinforcement learning with hierarchcies of machines. In *Neural Information Processing Systems 10*, 1998.

[13] Nathan Ratliff, J. Andrew Bagnell, and Martin Zinkevich. Maximum margin planning. In *Proceedings of the International Conference on Machine Learning*, 2006.

[14] Nathan Ratliff, David Bradley, J. Andrew Bagnell, and Joel Chestnutt. Boosting structured prediction for imitation learning. In *Neural Information Processing Systems 19*, 2007.

[15] Richard S. Sutton, Doina Precup, and Satinder Singh. Between mdps and semi-mdps: A framework for temporal abstraction in reinforcement learning. *Artificial Intelligence*, 112:181–211, 1999.

[16] Ben Taskar, Vassil Chatalbashev, Daphne Koller, and Carlos Guestrin. Learning structured prediction models: A large margin approach. In *Proceedings of the International Conference on Machine Learning*, 2005.

[17] I. Tsochantaridis, T. Joachims, T. Hofmann, and Y. Altun. Large margin methods for structured and interdependent output variables. *Journal of Machine Learning Research*, 6:1453–1484, 2005.

